# Maximal Margin Labeling for Multi-Topic Text Categorization

**Hideto Kazawa, Tomonori Izumitani, Hirotoshi Taira and Eisaku Maeda**
NTT Communication Science Laboratories
Nippon Telegraph and Telephone Corporation
2-4 Hikaridai, Seikacho, Sorakugun, Kyoto 619-0237 Japan
{kazawa,izumi,taira,maeda}@cslab.kecl.ntt.co.jp

## Abstract

In this paper, we address the problem of statistical learning for multi-topic text categorization (MTC), whose goal is to choose all relevant topics (*a label*) from a given set of topics. The proposed algorithm, Maximal Margin Labeling (MML), treats all possible labels as independent classes and learns a multi-class classifier on the induced multi-class categorization problem. To cope with the data sparseness caused by the huge number of possible labels, MML combines some prior knowledge about label prototypes and a maximal margin criterion in a novel way. Experiments with multi-topic Web pages show that MML outperforms existing learning algorithms including Support Vector Machines.

## 1 Multi-topic Text Categorization (MTC)

This paper addresses the problem of learning for multi-topic text categorization (MTC), whose goal is to select all topics relevant to a text from a given set of topics. In MTC, multiple topics may be relevant to a single text. We thus call a set of topics *label*, and say that a text is assigned a label, not a topic.

In almost all previous text categorization studies (e.g. [1, 2]), the label is predicted by judging each topic's relevance to the text. In this decomposition approach, the features specific to a *topic*, not a *label*, are regarded as important features. However, the approach may result in inefficient learning as we will explain in the following example.

Imagine an MTC problem of scientific papers where quantum computing papers are assigned multi-topic label "quantum physics (QP) & computer science (CS)". (QP and CS are topics in this example.) Since there are some words specific to quantum computing such as "qbit[1]", one can say that efficient MTC learners should use such words to assign label QP & CS. However, the decomposition approach is likely to ignore these words since they are only specific to a small portion of the whole QP or CS papers (there are many more QP and CS papers than quantum computing papers), and therefore are not discriminative features for either topic QP or CS.

| Symbol | Meaning |
|--------|---------|
| $\mathbf{x}(\in \mathbb{R}^d)$ | A document vector |
| $t_1, t_2, \ldots, t_l$ | Topics |
| $T$ | The set of all topics |
| $L, \lambda(\subset T)$ | A label |
| $L[j]$ | The binary representation of $L$. 1 if $t_j \in L$ and 0 otherwise. |
| $\Lambda(= 2^T)$ | The set of all possible labels |
| $\{(\mathbf{x}_i, L_i)\}_{i=1}^m$ | Training samples |

Table 1: Notation

Parametric Mixture Model (PMM) [3] adopts another approach to MTC. It is assumed in PMM that multi-topic texts are generated from a mixture of topic-specific word distributions. Its decision on labeling is done at once, not separately for each topic. However, PMM also has a problem with multi-topic specific features such as "qbit" since it is impossible for texts to have such features given PMM's mixture process.

These problems with multi-topic specific features are caused by dependency assumptions between labels, which are explicitly or implicitly made in existing methods. To solve these problems, we propose **Maximal Margin Labeling**, which treats **labels as independent classes** and learns a multi-class classifier on the induced multi-class problem.

In this paper, we first discuss why multi-class classifiers cannot be directly applied to MTC in Section 2. We then propose MML in Section 3, and address implementation issues in Section 4. In Section 5, MML is experimentally compared with existing methods using a collection of multi-topic Web pages. We summarize this paper in Section 6.

## 2 Solving MTC as a Multi-Class Categorization

To discuss why existing multi-class classifiers do not work in MTC, we start from the multi-class classifier proposed in [4]. Hereafter we use the notation given in Table 1. The multi-class classifier in [4] categorizes an object into the class whose prototype vector is the closest to the object's feature vector. By substituting label for class, the classifier can be written as follows.

$$f(\mathbf{x}) = \arg\max_{\lambda \in \Lambda} \langle \mathbf{x}, \mathbf{m}_\lambda \rangle_X \tag{1}$$

where $\langle, \rangle_X$ is the the inner product of $\mathbb{R}^d$, and $\mathbf{m}_\lambda \in \mathbb{R}^d$ is the prototype vector of label $\lambda$. Following the similar argument as in [4], the prototype vectors are learned by solving the following maximal margin problem[2].

$$\min_M \frac{1}{2}\|M\|^2 + C \sum_{1 \leq i \leq m} \sum_{\lambda \in \Lambda, \lambda \neq L_i} \xi_i^\lambda$$

$$\text{s.t.} \quad \langle \mathbf{x}_i, \mathbf{m}_{L_i} \rangle_X - \langle \mathbf{x}_i, \mathbf{m}_\lambda \rangle_X \geq 1 - \xi_i^\lambda \quad \text{for} \quad 1 \leq i \leq m, \forall \lambda \neq L_i, \tag{2}$$

where $M$ is the prototype matrix whose columns are the prototype vectors, and $\|M\|$ is the Frobenius matrix norm of $M$.

Note that Eq. (1) and Eq. (2) cover only training samples' labels, but also all possible labels. This is because the labels unseen in training samples may be relevant to test samples. In

usual multi-class problems, such unseen labels seldom exist. In MTC, however, the number of labels is generally very large (e.g. one of our datasets has 1,054 labels (Table 2)), and unseen labels often exist. Thus it is necessary to consider all possible labels in Eq. (1) and Eq. (2) since it is impossible to know which unseen labels are present in the test samples.

There are two problems with Eq. (1) and Eq. (2). The first problem is that they involve the prototype vectors of seldom or never seen labels. Without the help of prior knowledge about where the prototype vectors should be, it is impossible to obtain appropriate prototype vectors for such labels. The second problem is that these equations are computationally too demanding since they involve combinatorial maximization and summation over all possible labels, whose number can be quite large. (For example, the number is around $2^{30}$ in the datasets used in our experiments.) We will address the first problem in Section 3 and the second problem in Section 4.

## 3  Maximal Margin Labeling

In this section, we incorporate some prior knowledge about the location of prototype vectors into Eq. (1) and Eq. (2), and propose a novel MTC learning algorithm, **Maximal Margin Labeling (MML)**.

As prior knowledge, we simply assume that *the prototype vectors of similar labels should be placed close to each other.* Based on this assumption, we first rewrite Eq. (1) to yield

$$f(\mathbf{x}) = \arg\max_{\lambda \in \Lambda} \langle M^T \mathbf{x}, \mathbf{e}_\lambda \rangle_L, \tag{3}$$

where $\langle , \rangle_L$ is the inner product of $\mathbb{R}^{|\Lambda|}$ and $\{\mathbf{e}_\lambda\}_{\lambda \in \Lambda}$ is the orthonormal basis of $\mathbb{R}^{|\Lambda|}$. The classifier of Eq. (3) can be interpreted as a two-step process: the first step is to map the vector $\mathbf{x}$ into $\mathbb{R}^{|\Lambda|}$ by $M^T$, and the second step is to find the closest $\mathbf{e}_\lambda$ to image $M^T\mathbf{x}$. Then we replace $\{\mathbf{e}_\lambda\}_{\lambda \in \Lambda}$ with (generally) non-orthogonal vectors $\{\phi(\lambda)\}_{\lambda \in \Lambda}$ whose geometrical configuration reflects label similarity. More formally speaking, we use vectors $\{\phi(\lambda)\}_{\lambda \in \Lambda}$ that satisfy the condition

$$\langle \phi(\lambda_1), \phi(\lambda_2) \rangle_S = S(\lambda_1, \lambda_2) \quad \text{for } \forall \lambda_1, \lambda_2 \in \Lambda, \tag{4}$$

where $\langle , \rangle_S$ is an inner product of the vector space spanned by $\{\phi(\lambda)\}_{\lambda \in \Lambda}$, and $S$ is a Mercer kernel [5] on $\Lambda \times \Lambda$ and is a similarity measure between labels. We call the vector space spanned by $\{\phi(\lambda)\}$ $V_S$.

With this replacement, MML's classifier is written as follows.

$$f(\mathbf{x}) = \arg\max_{\lambda \in \Lambda} \langle W\mathbf{x}, \phi(\lambda) \rangle_S, \tag{5}$$

where $W$ is a linear map from $\mathbb{R}^d$ to $V_S$. $W$ is the solution of the following problem.

$$\min_W \quad \frac{1}{2}\|W\|^2 + C \sum_{i=1}^{m} \sum_{\lambda \in \Lambda, \lambda \neq L_i} \xi_i^\lambda$$

$$\text{s.t.} \quad \left\langle W\mathbf{x}_i, \frac{\phi(L_i) - \phi(\lambda)}{\|\phi(L_i) - \phi(\lambda)\|} \right\rangle \geq 1 - \xi_i^\lambda, \ \xi_i^\lambda \geq 0 \quad \text{for } 1 \leq i \leq m, \forall \lambda \neq L_i. \tag{6}$$

Note that if $\phi(\lambda)$ is replaced by $\mathbf{e}_\lambda$, Eq. (6) becomes identical to Eq. (2) except for a scale factor. Thus Eq. (5) and Eq. (6) are natural extensions of the multi-class classifier in [4]. We call the MTC classifier of Eq. (5) and Eq. (6) "Maximal Margin Labeling (MML)".

Figure 1 explains the margin (the inner product in Eq. (6)) in MML. The margin represents the distance from the image of the training sample $\mathbf{x}_i$ to the boundary between the correct label $L_i$ and wrong label $\lambda$. MML optimizes the linear map $W$ so that the smallest margin between all training samples and all possible labels becomes maximal, along with a penalty $C$ for the case that samples penetrate into the margin.

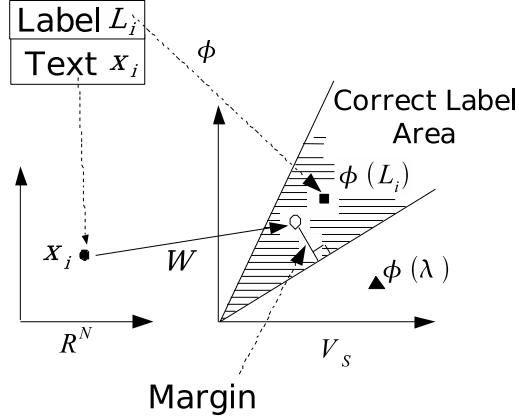

Figure 1: Maximal Margin Labeling

**Dual Form**  For numerical computation, the following Wolfe dual form of Eq. (6) is more convenient. (We omit its derivation due to space limits.)

$$\max_{\alpha_i^\lambda} \sum_{i,\lambda} \alpha_i^\lambda - \frac{1}{2} \sum_{i,\lambda} \sum_{i',\lambda'} \alpha_i^\lambda \alpha_{i'}^{\lambda'} (\mathbf{x}_i \cdot \mathbf{x}_{i'}) \frac{S(L_i, L_{i'}) - S(L_i, \lambda') - S(\lambda, L_{i'}) + S(\lambda, \lambda')}{2\sqrt{(1 - S(L_i, \lambda))(1 - S(L_{i'}, \lambda'))}}$$

$$\text{s.t.} \quad 0 \le \alpha_i^\lambda \le C \quad \text{for } 1 \le i \le m, \ \forall \lambda \neq L_i, \tag{7}$$

where we denote $\sum_{i=1}^m \sum_{\lambda \in \Lambda, \forall \lambda \neq L_i}$ by $\sum_{i,\lambda}$, and $\alpha_i^\lambda$ are the dual variables corresponding to the first inequality constraints in Eq. (6). Note that Eq. (7) does not contain $\phi(\lambda)$: all the computations involving $\phi$ can be done through the label similarity $S$. Additionally $\mathbf{x}_i$ only appears in the inner products, and therefore can be replaced by any kernel of $\mathbf{x}$.

Using the solution $\alpha_i^\lambda$ of Eq. (7), the MML's classifier in Eq. (5) can be written as follows.

$$f(\mathbf{x}) = \arg\max_{L \in \Lambda} \sum_{i,\lambda} \alpha_i^\lambda (\mathbf{x} \cdot \mathbf{x}_i) \frac{S(L_i, L) - S(\lambda, L)}{\sqrt{2(1 - S(L_i, \lambda))}}. \tag{8}$$

**Label Similarity**[3]  As examples of label similarity, we use two similarity measures: Dice measure and cosine measure.

Dice measure[4] $\quad S_D(\lambda_1, \lambda_2) = \dfrac{2|\lambda_1 \cap \lambda_2|}{|\lambda_1| + |\lambda_2|} = \dfrac{2\sum_{j=1}^l \lambda_1[j]\lambda_2[j]}{\sum_{j=1}^l \lambda_1[j] + \sum_{j=1}^l \lambda_2[j]}. \quad (9)$

Cosine measure $\quad S_C(\lambda_1, \lambda_2) = \dfrac{|\lambda_1 \cap \lambda_2|}{\sqrt{|\lambda_1|}\sqrt{|\lambda_2|}} = \dfrac{\sum_{j=1}^l \lambda_1[j]\lambda_2[j]}{\sqrt{\sum_{j=1}^l \lambda_1[j]}\sqrt{\sum_{j=1}^l \lambda_2[j]}} (10)$

## 4  Efficient Implementation

### 4.1  Approximation in Learning

Eq. (7) contains the sum over all possible labels. As the number of topics ($l$) increases, this summation rapidly becomes intractable since $|\Lambda|$ grows exponentially as $2^l$. To circumvent

this problem, we approximate the sum over all possible labels in Eq. (7) by the partial sum over $\alpha_i^\lambda$ of $|(A \cap B^c) \cup (A^c \cap B)| = 1$ and set all the other $\alpha_i^\lambda$ to zero. This approximation reduces the burden of the summation quite a lot: the number of summands is reduced from $2^l$ to $l$, which is a huge reduction especially when many topics exist.

To understand the rationale behind the approximation, first note that $\alpha_i^\lambda$ is the dual variable corresponding to the first inequality constraint (the margin constraint) in Eq. (7). Thus $\alpha_i^\lambda$ is non-zero if and only if $W\mathbf{x}_i$ falls in the margin between $\phi(L_i)$ and $\phi(\lambda)$. We assume that this margin violation mainly occurs when $\phi(\lambda)$ is "close" to $\phi(L_i)$, i.e. $|(A \cap B^c) \cup (A^c \cap B)| = 1$. If this assumption holds well, the proposed approximation of the sum will lead to a good approximation of the exact solution.

## 4.2   Polynomial Time Algorithms for Classification

The classification of MML (Eq. (8)) involves the combinatorial maximization over all possible labels, so it can be a computationally demanding process. However, efficient classification algorithms are available when either the cosine measure or dice measure is used as label similarity.

Eq. (8) can be divided into the subproblems by the number of topics in a label.

$$f(\mathbf{x}) \quad = \quad \underset{L \in \{\hat{L}_1, \hat{L}_2, \ldots, \hat{L}_l\}}{\arg \max} g(\mathbf{x}, L), \tag{11}$$

$$\hat{L}_n \quad = \quad \underset{L \in \Lambda, |L| \models n}{\arg \max} g(\mathbf{x}, L). \tag{12}$$

where $g(\mathbf{x})$ is

$$g(\mathbf{x}, L) \quad = \quad \sum_{j=1}^{l} c_n[j] L[j],$$

$$c_n[j] \quad = \quad \begin{cases} \sum_{i,\lambda} \frac{\alpha_i^\lambda (\mathbf{x} \cdot \mathbf{x}_i)}{\sqrt{2(1 - S_D(L_i, \lambda))}} \cdot \left( \frac{2 L_i[j]}{|L_i| + n} - \frac{2\lambda[j]}{|\lambda| + n} \right) & \text{if } S_D \text{ is used.} \\ \sum_{i,\lambda} \frac{\alpha_i^\lambda (\mathbf{x} \cdot \mathbf{x}_i)}{\sqrt{2(1 - S_C(L_i, \lambda))}} \left( \frac{L_i[j]}{\sqrt{|L_i|}\sqrt{n}} - \frac{\lambda[j]}{\sqrt{|\lambda|}\sqrt{n}} \right) & \text{if } S_C \text{ is used.} \end{cases} \tag{13}$$

Here $n = |L|$. The computational cost of Eq. (13) for all $j$ is $O(n_\alpha l)$ ($n_\alpha$ is the number of non-zero $\alpha$), and that of Eq. (12) is $O(l \log l)$. Thus the total cost of the classification by Eq. (11) is $O(n_\alpha l^2 + l^2 \log l)$. On the other hand, $n_\alpha$ is $O(ml)$ under the approximation described above. Therefore, the classification can be done within $O(ml^3)$ computational steps, which is a significant reduction from the case that the brute force search is used in Eq. (8).

# 5   Experiments

In this section, we report experiments that compared MML to PMM [3], SVM[5] [6], and BoosTexter [2] using a collection of Web pages. We used a normalized linear kernel $k(\mathbf{x}, \mathbf{x}') = \mathbf{x} \cdot \mathbf{x}' / \|\mathbf{x}\| \|\mathbf{x}'\|$ in MML and SVM. As for BoosTexter, "real abstaining Ad-aBoost.MH" was used as the weak learner.

## 5.1   Experimental Setup

The datasets used in our experiment represent the Web page collection used in [3] (Table 2). The Web pages were collected through the hyperlinks from Yahoo!'s top directory

| Dataset Name (Abbrev.) | #Text | #Voc | #Tpc | #Lbl | Label Size Frequency (%) | | | | |
|---|---|---|---|---|---|---|---|---|---|
| | | | | | 1 | 2 | 3 | 4 | ≥5 |
| Arts & Humanities (Ar) | 7,484 | 23,146 | 26 | 599 | 55.6 | 30.5 | 9.7 | 2.8 | 1.4 |
| Business & Economy (Bu) | 11,214 | 21,924 | 30 | 233 | 57.6 | 28.8 | 11.0 | 1.7 | 0.8 |
| Computers & Internet (Co) | 12,444 | 34,096 | 33 | 428 | 69.8 | 18.2 | 7.8 | 3.0 | 1.1 |
| Education (Ed) | 12,030 | 27,534 | 33 | 511 | 66.9 | 23.4 | 7.3 | 1.9 | 0.6 |
| Entertainment (En) | 12,730 | 32,001 | 21 | 337 | 72.3 | 21.1 | 4.5 | 1.0 | 1.1 |
| Health (He) | 9,205 | 30,605 | 32 | 335 | 53.2 | 34.0 | 9.5 | 2.4 | 0.9 |
| Recreation (Rc) | 12,828 | 30,324 | 22 | 530 | 69.2 | 23.1 | 5.6 | 1.4 | 0.6 |
| Reference (Rf) | 8,027 | 39,679 | 33 | 275 | 85.5 | 12.6 | 1.5 | 0.3 | 0.1 |
| Science (Si) | 6,428 | 37,187 | 40 | 457 | 68.0 | 22.3 | 7.3 | 1.9 | 0.5 |
| Social Science (SS) | 12,111 | 52,350 | 39 | 361 | 78.4 | 17.0 | 3.7 | 0.7 | 0.3 |
| Society & Culture (SC) | 14,512 | 31,802 | 27 | 1054 | 59.6 | 26.1 | 9.2 | 2.9 | 2.2 |

Table 2: A summary of the web page datasets. "#Text" is the number of texts in the dataset, "#Voc" the number of vocabularies (i.e. features), "#Tpc" the number of topics, "#Lbl" the number of labels, and "Label Size Frequency" is the relative frequency of each label size. (Label size is the number of topics in a label.)

| Method | Feature Type | Parameter |
|---|---|---|
| MML | TF, TF×IDF | $C = 0.1, \underline{1}, 10$ |
| PMM | TF | Model1, Model2 |
| SVM | TF, TF×IDF | $C = 0.1, \underline{1}, 10$ |
| Boost | Binary | $R = \{2, 4, 6, \underline{8}, 10\} \times 10^3$ |

Table 3: Candidate feature types and learning parameters. ($R$ is the number of weak hypotheses.) The underlined fetures and parameters were selected for the evaluation with the test data.

(www.yahoo.com), and then divided into 11 datasets by Yahoo's top category. Each page is labeled with the Yahoo's *second* level sub-categories from which the page is hyperlinked. (Thus, the sub-categories are *topics* in our term.) See [3] for more details about the collection. Then the Web pages were converted into three types of feature vectors: (a) Binary vectors, where each feature indicates the presence (absence) of a term by 1 (0); (b) TF vectors, where each feature is the number of appearances of a term (term frequency); and (c) TF×IDF vectors, where each feature is the product of term frequency and inverse document frequency [7].

To select the best combinations of feature types and learning parameters such as the penalty $C$ for MML, the learners were trained on 2,000 Web pages with all combinations of feature and parameter listed in Table 3, and then were evaluated by labeling F-measure on independently drawn development data. The combinations which achieve the best labeling F-measures (underlined in Table 3) were used in the following experiments.

## 5.2 Evaluation Measures

We used three measures to evaluate labeling performance: labeling F-measure, exact match ratio, and retrieval F-measure. In the following definitions, $\{L_i^{\mathrm{pred}}\}_{i=1}^n$ and $\{L_i^{\mathrm{true}}\}_{i=1}^n$ mean the predicted labels and the true labels, respectively.

**Labeling F-measure**  Labeling F-measure $F_L$ evaluates the average labeling performance while taking partial match into account.

$$F_L = \frac{1}{n} \sum_{i=1}^n \frac{2|L_i^{\mathrm{pred}} \cap L_i^{\mathrm{true}}|}{|L_i^{\mathrm{pred}}| + |L_i^{\mathrm{true}}|} = \frac{1}{n} \sum_{i=1}^n \frac{2\sum_{j=1}^l L_i^{\mathrm{pred}}[j] L_i^{\mathrm{true}}[j]}{\sum_{j=1}^l (L_i^{\mathrm{pred}}[j] + L_i^{\mathrm{true}}[j])}. \qquad (14)$$

| Data-set | Labeling F-measure | | | | | Exact Match Ratio | | | | | Retrieval F-measure | | | | |
|---|---|---|---|---|---|---|---|---|---|---|---|---|---|---|---|
| | MD | MC | PM | SV | BO | MD | MC | PM | SV | BO | MD | MC | PM | SV | BO |
| Ar | **0.55** | 0.44 | _0.50_ | 0.46 | 0.38 | **0.44** | _0.32_ | 0.21 | 0.29 | 0.22 | **0.30** | 0.26 | 0.24 | _0.29_ | 0.22 |
| Bu | 0.80 | **0.81** | 0.75 | 0.76 | 0.75 | **0.63** | _0.62_ | 0.48 | 0.57 | 0.53 | 0.25 | _0.27_ | 0.20 | **0.29** | 0.20 |
| Co | **0.62** | 0.59 | _0.61_ | 0.55 | 0.47 | **0.51** | _0.46_ | 0.35 | 0.41 | 0.34 | _0.27_ | 0.25 | 0.19 | **0.30** | 0.17 |
| Ed | **0.56** | 0.43 | _0.51_ | 0.48 | 0.37 | **0.45** | _0.34_ | 0.19 | 0.30 | 0.23 | **0.25** | 0.23 | 0.21 | **0.25** | 0.16 |
| En | **0.64** | 0.52 | _0.61_ | 0.54 | 0.49 | **0.55** | _0.44_ | 0.31 | 0.42 | 0.36 | **0.37** | 0.33 | 0.30 | _0.35_ | 0.29 |
| He | _0.74_ | **0.74** | 0.66 | 0.67 | 0.60 | **0.58** | _0.53_ | 0.34 | 0.47 | 0.39 | **0.35** | **0.35** | 0.23 | **0.35** | 0.26 |
| Rc | **0.63** | 0.46 | _0.55_ | 0.49 | 0.44 | **0.54** | _0.38_ | 0.25 | 0.37 | 0.33 | **0.47** | 0.39 | 0.36 | _0.40_ | 0.33 |
| Rf | **0.67** | 0.58 | _0.63_ | 0.56 | 0.50 | **0.60** | _0.51_ | 0.39 | 0.49 | 0.41 | **0.29** | _0.25_ | 0.24 | _0.25_ | 0.16 |
| Si | **0.61** | _0.54_ | 0.52 | 0.47 | 0.39 | **0.52** | _0.43_ | 0.22 | 0.36 | 0.28 | **0.37** | _0.35_ | 0.28 | 0.31 | 0.19 |
| SS | **0.73** | _0.71_ | 0.66 | 0.64 | 0.59 | **0.65** | _0.60_ | 0.45 | 0.55 | 0.49 | **0.36** | _0.35_ | 0.18 | 0.31 | 0.15 |
| SC | **0.60** | _0.55_ | 0.54 | 0.49 | 0.44 | **0.44** | _0.40_ | 0.21 | 0.32 | 0.27 | **0.29** | _0.28_ | 0.25 | 0.26 | 0.20 |
| Avg | **0.65** | 0.58 | _0.59_ | 0.56 | 0.49 | **0.54** | _0.46_ | 0.31 | 0.41 | 0.35 | **0.32** | 0.30 | 0.24 | _0.31_ | 0.21 |

Table 4: The performance comparison by labeling F-measure (left), exact match ratio (middle) and retrieval F-measure (right). The **bold figures** are the best ones among the five methods, and the underlined figures the second best ones. MD, MC, PM, SV, and BO represent MML with $S_D$, MML with $S_C$, PMM, SVM and BoosTexter, respectively.

**Exact Match Ratio**  Exact match ratio $EX$ counts only exact matches between the predicted label and the true label.

$$EX = \frac{1}{n} \sum_{i=1}^{n} I[L_i^{\text{pred}} = L_i^{\text{true}}], \qquad (15)$$

where $I[S]$ is 1 if the statement $S$ is true and 0 otherwise.

**Retrieval F-measure**[6]  For real tasks, it is also important to evaluate retrieval performance, i.e. how accurately classifiers can find relevant texts for a given topic. Retrieval F-measure $F_R$ measures the average retrieval performance over all topics.

$$F_R = \frac{1}{l} \sum_{j=1}^{l} \frac{2 \sum_{i=1}^{n} L_i^{\text{pred}}[j] L_i^{\text{true}}[j]}{\sum_{i=1}^{n} (L_i^{\text{pred}}[j] + L_i^{\text{true}}[j])}. \qquad (16)$$

## 5.3  Results

First we trained the classifiers with randomly chosen 2,000 samples. We then calculated the three evaluation measures for 3,000 other randomly chosen samples. This process was repeated five times, and the resulting averaged values are shown in Table 4. Table 4 shows that the MMLs with Dice measure outperform other methods in labeling F-measure and exact match ratio. The MMLs also show the best performance with regard to retrieval F-measure although the margins to the other methods are not as large as observed in labeling F-measure and exact match ratio. Note that no classifier except MML with Dice measure achieves good labeling on all the three measures. For example, PMM shows high labeling F-measures, but its performance is rather poor when evaluated in retrieval F-measure.

As the second experiment, we evaluated the classifiers trained with 250–2000 training samples on the same test samples. Figure 2 shows each measure averaged over all datasets. It is observed that the MMLs show high generalization even when training data is small. An interesting point is that MML with cosine measure achieves rather high labeling F-measures and retrieval F-measure with training data of smaller size. Such high-performace, however, does not continue when trained on larger data.

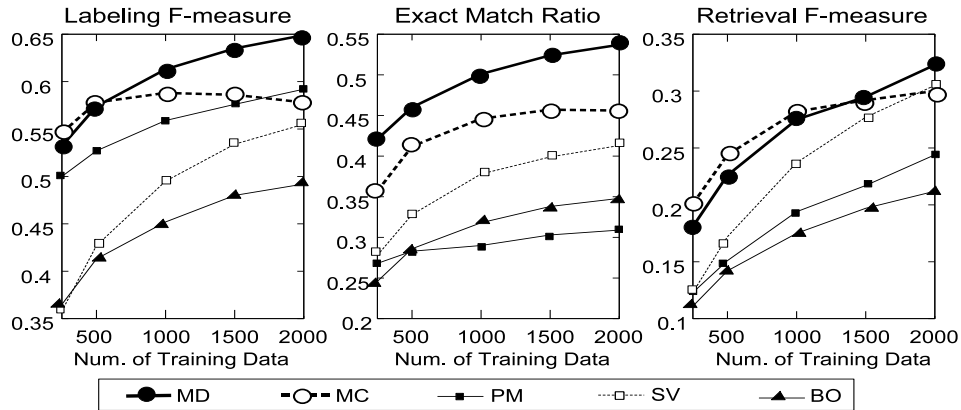

Figure 2: The learning curve of labeling F-measure (left), exact match ratio (middle) and retrieval F-measure (right). MD, MC, PM, SV, BO mean the same as in Table 4.

## 6 Conclusion

In this paper, we proposed a novel learning algorithm for multi-topic text categorization. The algorithm, Maximal Margin Labeling, embeds labels (sets of topics) into a similarity-induced vector space, and learns a large margin classifier in the space. To overcome the demanding computational cost of MML, we provide an approximation method in learning and efficient classification algorithms. In experiments on a collection of Web pages, MML outperformed other methods including SVM and showed better generalization.

## Acknowledgement

The authors would like to thank Naonori Ueda, Kazumi Saito and Yuji Kaneda of Nippon Telegraph and Telephone Corporation for providing PMM's codes and the datasets.

## Footnotes

[1]Qbit is a unit of quantum information, and frequently appears in real quantum computing literatures, but rarely seen in other literatures.

[2]In Eq.(2), we penalize all violation of the margin constraints. On the other hand, Crammer and Singer penalize only the largest violation of the margin constraint for each training sample [4]. We chose the "penalize-all" approach since it leads to an optimization problem without equality constraints (see Eq.(7)), which is much easier to solve than the one in [4].

[3]The following discussion is easily extended to include the case that both $\lambda_1$ and $\lambda_2$ are empty although we do not discuss the case due to space limits.

[5]For each topic, an SVM classifier is trained to predict whether the topic is relevant (positive) or irrelevant (negative) to input doucments.

[6]$F_R$ is called "the macro average of F-measures" in the text categorization community.

## References

[1] Thorsten Joachims. Text categorization with support vector machines: learning with many relevant features. In Claire Nédellec and Céline Rouveirol, editors, *Proc. of the 10th European Conference on Machine Learning*, number 1398, pages 137–142, 1998.

[2] Robert E. Schapire and Yoram Singer. BoosTexter: A boosting-based system for text categorization. *Machine Learning*, 39(2/3):135–168, 2000.

[3] Naonori Ueda and Kazumi Saito. Parametoric mixture models for multi-topic text. In *Advances in Neural Information Processing Systems 15*, pages 1261–1268, 2003.

[4] Koby Crammer and Yoram Singer. On the algorithmic implementation of multiclass kernel-based vector machines. *Journal of Machine Learning Research*, 2:265–292, 2001.

[5] Klaus-Robert Müller, Sebastian Mika, Gunnar Rätsch, Koji Tsuda, and Bernhard Schölkopf. An introduction to kernel-based learning algorithms. *IEEE Transactions on Neural Networks*, 12(2):181–201, 2001.

[6] Vladimir N. Vapnik. *Statistical Learning Theory*. John Wiley & Sons, Inc., 1998.

[7] Ricardo Baeza-Yates and Berthier Ribeiro-Neto. *Modern Information Retrieval*. Addison-Wealy, 1999.
